# An Adaptive Metric Machine for Pattern Classification

**Carlotta Domeniconi, Jing Peng[+], Dimitrios Gunopulos**
Dept. of Computer Science, University of California, Riverside, CA 92521
+ Dept. of Computer Science, Oklahoma State University, Stillwater, OK 74078
{*carlotta,dg*}*@cs.ucr.edu, jpeng@cs.okstate.edu*

## Abstract

Nearest neighbor classification assumes locally constant class conditional probabilities. This assumption becomes invalid in high dimensions with finite samples due to the curse of dimensionality. Severe bias can be introduced under these conditions when using the nearest neighbor rule. We propose a locally adaptive nearest neighbor classification method to try to minimize bias. We use a *Chi-squared* distance analysis to compute a flexible metric for producing neighborhoods that are elongated along less relevant feature dimensions and constricted along most influential ones. As a result, the class conditional probabilities tend to be smoother in the modified neighborhoods, whereby better classification performance can be achieved. The efficacy of our method is validated and compared against other techniques using a variety of real world data.

## 1 Introduction

In classification, a feature vector $\mathbf{x} = (x_1, \cdots, x_q)^t \in \Re^q$, representing an object, is assumed to be in one of $J$ classes $\{i\}_{i=1}^J$, and the objective is to build classifier machines that assign $\mathbf{x}$ to the correct class from a given set of $N$ training samples.

The $K$ nearest neighbor (NN) classification method [3, 5, 7, 8, 9] is a simple and appealing approach to this problem. Such a method produces continuous and overlapping, rather than fixed, neighborhoods and uses a different neighborhood for each individual query so that all points in the neighborhood are close to the query, to the extent possible. In addition, it has been shown [4, 6] that the one NN rule has asymptotic error rate that is at most twice the Bayes error rate, independent of the distance metric used.

The NN rule becomes less appealing in finite training samples, however. This is due to the curse-of-dimensionality [2]. Severe bias can be introduced in the NN rule in a high dimensional input feature space with finite samples. As such, the choice of a distance measure becomes crucial in determining the outcome of nearest neighbor classification. The commonly used Euclidean distance measure, while simple computationally, implies that the input space is isotropic or homogeneous. However, the assumption for isotropy is often invalid and generally undesirable in many practical applications. In general, distance computation does not vary

with equal strength or in the same proportion in all directions in the feature space emanating from the input query. Capturing such information, therefore, is of great importance to any classification procedure in high dimensional settings.

In this paper we propose an adaptive metric classification method to try to minimize bias in high dimensions. We estimate a flexible metric for computing neighborhoods based on *Chi-squared* distance analysis. The resulting neighborhoods are highly adaptive to query locations. Moreover, the neighborhoods are elongated along less relevant feature dimensions and constricted along most influential ones. As a result, the class conditional probabilities tend to be constant in the modified neighborhoods, whereby better classification performance can be obtained.

## 2  Local Feature Relevance Measure

Our technique is motivated as follows. Let $\mathbf{x}_0$ be the test point whose class membership we are predicting. In the one NN classification rule, a single nearest neighbor $\mathbf{x}$ is found according to a distance metric $D(\mathbf{x}, \mathbf{x}_0)$. Let $p(j|\mathbf{x})$ be the class conditional probability at point $\mathbf{x}$. Consider the weighted *Chi-squared* distance [8, 11]

$$D(\mathbf{x}, \mathbf{x}_0) = \sum_{j=1}^{J} \frac{[\Pr(j|\mathbf{x}) - \Pr(j|\mathbf{x}_0)]^2}{\Pr(j|\mathbf{x}_0)}, \tag{1}$$

which measures the distance between $\mathbf{x}_0$ and the point $\mathbf{x}$, in terms of the difference between the class posterior probabilities at the two points. Small $D(\mathbf{x}, \mathbf{x}_0)$ indicates that the classification error rate will be close to the asymptotic error rate for one nearest neighbor. In general, this can be achieved when $\Pr(j|\mathbf{x}) = \Pr(j|\mathbf{x}_0)$, which states that if $\Pr(j|\mathbf{x})$ can be sufficiently well approximated at $\mathbf{x}_0$, the asymptotic 1-NN error rate might result in finite sample settings.

Equation (1) computes the distance between the true and estimated posteriors. Now, imagine we replace $\Pr(j|\mathbf{x}_0)$ with a quantity that attempts to predict $\Pr(j|\mathbf{x})$ under the constraint that the quantity is conditioned at a location along a particular feature dimension. Then, the *Chi-squared* distance (1) tells us the extent to which that dimension can be relied on to predict $\Pr(j|\mathbf{x})$. Thus, Equation (1) provides us with a foundation upon which to develop a theory of feature relevance in the context of pattern classification.

Based on the above discussion, our proposal is the following. We first notice that $\Pr(j|\mathbf{x})$ is a function of $\mathbf{x}$. Therefore, we can compute the conditional expectation of $p(j|\mathbf{x})$, denoted by $\overline{\Pr}(j|x_i = z)$, given that $x_i$ assumes value $z$, where $x_i$ represents the $i$th component of $\mathbf{x}$. That is, $\overline{\Pr}(j|x_i = z) = E[\Pr(j|\mathbf{x})|x_i = z] = \int \Pr(j|\mathbf{x})p(\mathbf{x}|x_i = z)d\mathbf{x}$. Here $p(\mathbf{x}|x_i = z)$ is the conditional density of the other input variables. Let

$$r_i(\mathbf{x}) = \sum_{j=1}^{J} \frac{[\Pr(j|\mathbf{x}) - \overline{\Pr}(j|x_i = z_i)]^2}{\overline{\Pr}(j|x_i = z_i)}. \tag{2}$$

$r_i(\mathbf{x})$ represents the ability of feature $i$ to predict the $\Pr(j|\mathbf{x})$s at $x_i = z_i$. The closer $\overline{\Pr}(j|x_i = z_i)$ is to $\Pr(j|\mathbf{x})$, the more information feature $i$ carries for predicting the class posterior probabilities locally at $\mathbf{x}$.

We can now define a measure of feature relevance for $\mathbf{x}_0$ as

$$\bar{r}_i(\mathbf{x}_0) = \frac{1}{K} \sum_{\mathbf{z} \in N(\mathbf{x}_0)} r_i(\mathbf{z}), \tag{3}$$

where $N(\mathbf{x}_0)$ denotes the neighborhood of $\mathbf{x}_0$ containing the $K$ nearest training points, according to a given metric. $\bar{r}_i$ measures how well on average the class posterior probabilities can be approximated along input feature $i$ within a local neighborhood of $\mathbf{x}_0$. Small $\bar{r}_i$ implies that the class posterior probabilities will be well captured along dimension $i$ in the vicinity of $\mathbf{x}_0$. Note that $\bar{r}_i(\mathbf{x}_0)$ is a function of both the test point $\mathbf{x}_0$ and the dimension $i$, thereby making $\bar{r}_i(\mathbf{x}_0)$ a local relevance measure.

The relative relevance, as a weighting scheme, can then be given by the following exponential weighting scheme

$$w_i(\mathbf{x}_0) = \exp(cR_i(\mathbf{x}_0))/\sum_{l=1}^{q}\exp(cR_l(\mathbf{x}_0)) \tag{4}$$

where $c$ is a parameter that can be chosen to maximize (minimize) the influence of $\bar{r}_i$ on $w_i$, and $R_i(\mathbf{x}) = \max_j \bar{r}_j(\mathbf{x}) - \bar{r}_i(\mathbf{x})$. When $c = 0$ we have $w_i = 1/q$, thereby ignoring any difference between the $\bar{r}_i$'s. On the other hand, when $c$ is large a change in $\bar{r}_i$ will be exponentially reflected in $w_i$. In this case, $w_i$ is said to follow the Boltzmann distribution. The exponential weighting is more sensitive to changes in local feature relevance (3) and gives rise to better performance improvement. Thus, (4) can be used as weights associated with features for weighted distance computation $D(\mathbf{x},\mathbf{y}) = \sqrt{\sum_{i=1}^{q} w_i(x_i - y_i)^2}$. These weights enable the neighborhood to elongate less important feature dimensions, and, at the same time, to constrict the most influential ones. Note that the technique is *query-based* because weightings depend on the query [1].

## 3   Estimation

Since both $\Pr(j|\mathbf{x})$ and $\overline{\Pr}(j|x_i = z_i)$ in (3) are unknown, we must estimate them using the training data $\{\mathbf{x}_n, y_n\}_{n=1}^{N}$ in order for the relevance measure (3) to be useful in practice. Here $y_n \in \{1, \cdots, J\}$. The quantity $\Pr(j|\mathbf{x})$ is estimated by considering a neighborhood $N_1(\mathbf{x})$ centered at $\mathbf{x}$:

$$\hat{\Pr}(j|\mathbf{x}) = \frac{\sum_{n=1}^{N} 1(\mathbf{x}_n \in N_1(\mathbf{x}))1(y_n = j)}{\sum_{n=1}^{N} 1(\mathbf{x}_n \in N_1(\mathbf{x}))}, \tag{5}$$

where $1(\cdot)$ is an indicator function such that it returns 1 when its argument is true, and 0 otherwise.

To compute $\overline{\Pr}(j|x_i = z) = E[\Pr(j|\mathbf{x})|x_i = z]$, we introduce a dummy variable $g_j$ such that if $y = j$, then $g_j|\mathbf{x} = 1$, otherwise $g_j|\mathbf{x} = 0$, where $j = 1, \cdots, J$. We then have $\Pr(j|\mathbf{x}) = E[g_j|\mathbf{x}]$, from which it is not hard to show that $\overline{\Pr}(j|x_i = z) = E[g_j|x_i = z]$. However, since there may not be any data at $x_i = z$, the data from the neighborhood of $x$ along dimension $i$ are used to estimate $E[g_j|x_i = z]$, a strategy suggested in [7]. In detail, by noticing $g_j = 1(y = j)$ the estimate can be computed from

$$\hat{\overline{\Pr}}(j|x_i = z_i) = \frac{\sum_{\mathbf{x}_n \in N_2(\mathbf{x})} 1(|x_{ni} - x_i| \leq \Delta_i)1(y_n = j)}{\sum_{\mathbf{x}_n \in N_2(\mathbf{x})} 1(|x_{ni} - x_i| \leq \Delta_i)}, \tag{6}$$

where $N_2(\mathbf{x})$ is a neighborhood centered at $\mathbf{x}$ (larger than $N_1(\mathbf{x})$), and the value of $\Delta_i$ is chosen so that the interval contains a fixed number $L$ of points: $\sum_{n=1}^{N} 1(|x_{ni} - x_i| \leq \Delta_i)1(\mathbf{x}_n \in N_2(\mathbf{x})) = L$. Using the estimates in (5) and in (6), we obtain an empirical measure of the relevance (3) for each input variable $i$.

# 4 Empirical Results

In the following we compare several classification methods using real data: (1) Adaptive metric nearest neighbor (ADAMENN) method (one iteration) described above, coupled with the exponential weighting scheme (4); (2) i-ADAMENN - ADAMENN with five iterations; (3) Simple K-NN method using the Euclidean distance measure; (4) C4.5 decision tree method [12]; (5) Machete [7] - an adaptive NN procedure, in which the input variable used for splitting at each step is the one that maximizes the estimated local relevance (7); (6) Scythe [7] - a generalization of the Machete algorithm, in which the input variables influence each split in proportion to their estimated local relevance, rather than the winner-take-all strategy of Machete; (7) DANN - discriminant adaptive nearest neighbor classification [8]; and (8) i-DANN - DANN with five iterations [8].

In all the experiments, the features are first normalized over the training data to have zero mean and unit variance, and the test data are normalized using the corresponding training mean and variance. Procedural parameters for each method were determined empirically through cross-validation.

Table 1: Average classification error rates.

|  | Iris | Sonar | Vowel | Glass | Image | Seg | Letter | Liver | Lung |
|---|---|---|---|---|---|---|---|---|---|
| ADAMENN | 3.0 | 9.1 | 10.7 | 24.8 | 5.2 | 2.4 | 5.1 | 30.7 | 40.6 |
| i-ADAMENN | 5.0 | 9.6 | 10.9 | 24.8 | 5.2 | 2.5 | 5.3 | 30.4 | 40.6 |
| K-NN | 6.0 | 12.5 | 11.8 | 28.0 | 6.1 | 3.6 | 6.9 | 32.5 | 50.0 |
| C4.5 | 8.0 | 23.1 | 36.7 | 31.8 | 21.6 | 3.7 | 16.4 | 38.3 | 59.4 |
| Machete | 5.0 | 21.2 | 20.2 | 28.0 | 12.3 | 3.2 | 9.1 | 27.5 | 50.0 |
| Scythe | 4.0 | 16.3 | 15.5 | 27.1 | 5.0 | 3.3 | 7.2 | 27.5 | 50.0 |
| DANN | 6.0 | 7.7 | 12.5 | 27.1 | 12.9 | 2.5 | 3.1 | 30.1 | 46.9 |
| i-DANN | 6.0 | 9.1 | 21.8 | 26.6 | 18.1 | 3.7 | 6.1 | 27.8 | 40.6 |

**Classification Data Sets.** The data sets used were taken from the UCI Machine Learning Database Repository [10], except for the unreleased *image* data set. They are: 1. **Iris data**. This data set consists of $q = 4$ measurements made on each of $N = 100$ iris plants of $J = 2$ species; 2. **Sonar data**. This data set consists of $q = 60$ frequency measurements made on each of $N = 208$ data of $J = 2$ classes ("mines" and "rocks"); 3. **Vowel data**. This example has $q = 10$ measurements and 11 classes. There are total of $N = 528$ samples in this example; 4. **Glass data**. This data set consists of $q = 9$ chemical attributes measured for each of $N = 214$ data of $J = 6$ classes; 5. **Image data**. This data set consists of 40 texture images that are manually classified into 15 classes. The number of images in each class varies from 16 to 80. The images in this database are represented by $q = 16$ dimensional feature vectors; 6. **Seg data**. This data set consists of images that were drawn randomly from a database of 7 outdoor images. There are $J = 7$ classes, each of which has 330 instances. Thus, there are $N = 2,310$ images in the database. These images are represented by $q = 19$ real valued attributes; 7. **Letter data**. This data set consists of $q = 16$ numerical attributes and $J = 26$ classes; 8. **Liver data**. This data set consists of 345 instances, represented by $q = 6$ numerical attributes, and $J = 2$ classes; and 9. **Lung data**. This example has 32 instances having $q = 56$ numerical features and $J = 3$ classes.

**Results:** Table 1 shows the (cross-validated) error rates for the eight methods under consideration on the nine real data sets. Note that the average error rates

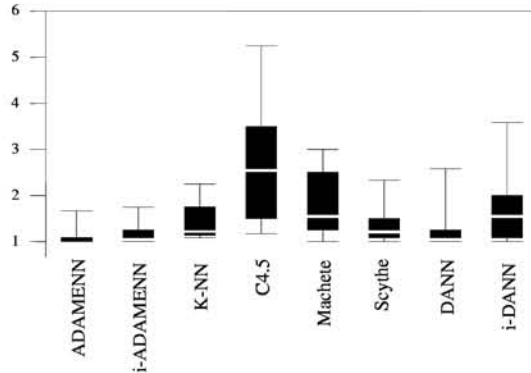

Figure 1: Performance distributions.

for the Iris, Sonar, Glass, Liver and Lung data sets were based on leave-one-out cross-validation, whereas the error rates for the Vowel and Image data were based on ten two-fold cross-validation, and two ten-fold cross-validation for the Seg and Letter data, since larger data sets are available in these four cases.

Table 1 shows clearly that ADAMENN achieved the best or near best performance over the nine real data sets, followed by i-ADAMENN. It seems natural to ask the question of robustness. That is, how well a particular method $m$ performs on average in situations that are most favorable to other procedures. Following Friedman [7], we capture robustness by computing the ratio $b_m$ of its error rate $e_m$ and the smallest error rate over all methods being compared in a particular example: $b_m = e_m / \min_{1 \leq k \leq 8} e_k$. Thus, the best method $m^*$ for that example has $b_{m^*} = 1$, and all other methods have larger values $b_m \geq 1$, for $m \neq m^*$. The larger the value of $b_m$, the worse the performance of the $mth$ method is in relation to the best one for that example, among the methods being compared. The distribution of the $b_m$ values for each method $m$ over all the examples, therefore, seems to be a good indicator of robustness.

Fig. 1 plots the distribution of $b_m$ for each method over the nine data sets. The dark area represents the lower and upper quartiles of the distribution that are separated by the median. The outer vertical lines show the entire range of values for the distribution. It is clear that the most robust method over the data sets is ADAMENN. In 5/9 of the data its error rate was the best (median = 1.0). In 8/9 of them it was no worse than 18% higher than the best error rate. In the worst case it was 65%. In contrast, C4.5 has the worst distribution, where the corresponding numbers are 267%, 432% and 529%.

**Bias and Variance Calculations:** For a two-class problem with $\Pr(Y = 1|\mathbf{x}) = p(\mathbf{x})$, we compute a nearest neighborhood at a query $\mathbf{x}_0$ and find the nearest neighbor $\mathbf{X}$ having class label $Y(\mathbf{X})$ (random variable). The estimate of $p(\mathbf{x}_0)$ is $Y(\mathbf{X})$. The bias and variance of $Y(\mathbf{X})$ are: $Bias = Ep(\mathbf{X}) - p(\mathbf{x}_0)$ and $Var = Ep(\mathbf{X})(1 - Ep(\mathbf{X}))$, where the expectation is computed over the distribution of the nearest neighbor $\mathbf{X}$ [8].

We performed simulations to estimate the bias and variance of ADAMENN, KNN, DANN and Machete on the following two-class problem. There are $q = 2$ input features and 180 training data. Each class contains three spherical bivariate normal subclasses, having standard deviation 0.75. The means of the 6 subclasses are

chosen at random without replacement from the integers $[1, 2, \ldots, 8] \times [1, 2, \ldots, 8]$. For each class, data are evenly drawn from each of the normal subclasses. Fig. 2 shows the bias and variance estimates from each method at locations $(5, 5, 0, \cdots, 0)$ and $(2.3, 7, 0, \cdots, 0)$, as a function of the number of noise variables over five independently generated training sets. Here the noise variables have independent standard Gaussian distributions. The true probability of class 1 for $(5, 5, 0, \cdots, 0)$ and $(2.3, 7, 0, \cdots, 0)$ are 0.943 and 0.747, respectively. The four methods have similar variance, since they all use three neighbors for classification. While the bias of KNN and DANN increases with increasing number of noise variables, ADAMENN retains a low bias by averaging out noise.

## 5  Related Work

Friedman [7] describes an approach to learning local feature relevance that recursively homes in on a query along the most (locally) relevant dimension, where local relevance is computed from a reduction in prediction error given the query's value along that dimension. This method performs well on a number of classification tasks. In our notations, local relevance can be described by

$$I_i^2(\mathbf{x}) = \sum_{j=1}^{J} (\overline{\mathrm{Pr}}(j) - \overline{\mathrm{Pr}}(j|x_i = z_i)])^2, \qquad (7)$$

where $\overline{\mathrm{Pr}}(j)$ represents the expected value of $\mathrm{Pr}(j|\mathbf{x})$. In this case, the most informative dimension is the one that deviates the most from $\overline{\mathrm{Pr}}(j)$.

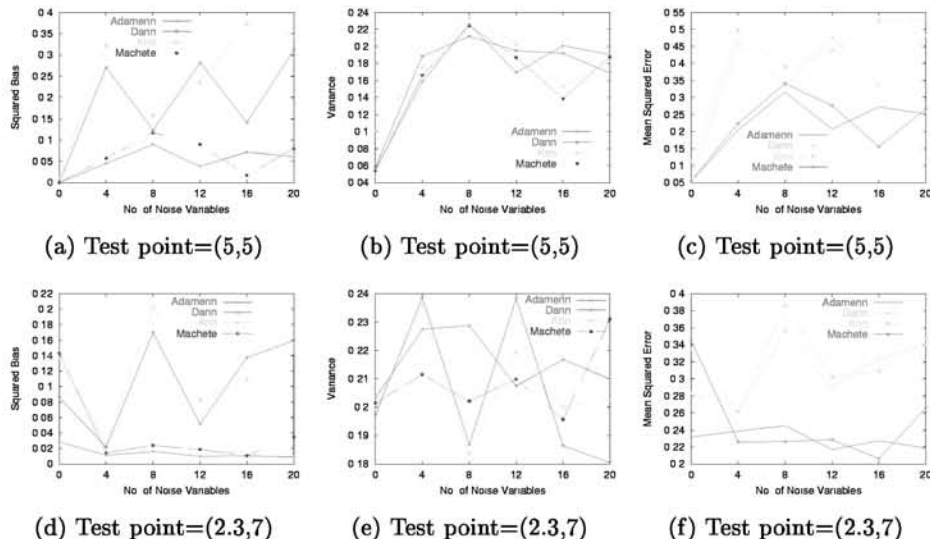

Figure 2: Bias and variance estimates.

The main difference, however, between our relevance measure (3) and Friedman's (7) is the first term in the squared difference. While the class conditional probability is used in our relevance measure, its expectation is used in Friedman's. As a result, a feature dimension is more relevant than others when it minimizes (2) in case of our relevance measure, whereas it maximizes (7) in case of Friedman's. Furthermore, we

take into account not only the test point $\mathbf{x}_0$ itself, but also its $K$ nearest neighbors, resulting in a relevance measure (3) that is often more robust.

In [8], Hastie and Tibshirani propose an adaptive nearest neighbor classification method based on linear discriminant analysis. The method computes a distance metric as a product of properly weighted within and between sum of squares matrices. They show that the resulting metric approximates the *Chi-squared* distance (1) by a Taylor series expansion. While sound in theory, the method has limitations. The main concern is that in high dimensions we may never have sufficient data to fill in $q \times q$ matrices. It is interesting to note that our work can serve as a potential bridge between Friedman's and that of Hastie and Tibshirani.

## 6   Summary and Conclusions

This paper presents an adaptive metric method for effective pattern classification. This method estimates a flexible metric for producing neighborhoods that are elongated along less relevant feature dimensions and constricted along most influential ones. As a result, the class conditional probabilities tend to be more homogeneous in the modified neighborhoods. The experimental results show clearly that the ADAMENN algorithm can potentially improve the performance of K-NN and recursive partitioning methods in some classification problems, especially when the relative influence of input features changes with the location of the query to be classified in the input feature space. The results are also in favor of ADAMENN over similar competing methods such as Machete and DANN.

## References

[1] Atkeson, C., Moore, A.W., and Schaal, S. (1997). "Locally Weighted Learning," *AI Review*. 11:11-73.

[2] Bellman, R.E. (1961). *Adaptive Control Processes*. Princeton Univ. Press.

[3] Cleveland, W.S. and Devlin, S.J. (1988). "Locally Weighted Regression: An Approach to Regression Analysis by Local Fitting," *J. Amer. Statist. Assoc.* **83**, 596-610.

[4] Cover, T.M. and Hart, P.E. (1967). "Nearest Neighbor Pattern Classification," *IEEE Trans. on Information Theory*, pp. 21-27.

[5] Domeniconi, C., Peng, J., and Gunopulos, D. (2000). "Adaptive Metric Nearest Neighbor Classification," Proc. of IEEE Conf. on CVPR, pp. 517-522, Hilton Head Island, South Carolina.

[6] Duda, R.O. and Hart, P.E. (1973). *Pattern Classification and Scene Analysis*. John Wiley & Sons, Inc..

[7] Friedman, J.H. (1994). "Flexible Metric Nearest Neighbor Classification," Tech. Report, Dept. of Statistics, Stanford University.

[8] Hastie, T. and Tibshirani, R. (1996). "Discriminant Adaptive Nearest Neighbor Classification", *IEEE Trans. on Pattern Analysis and Machine Intelligence*, Vol. 18, No. 6, pp. 607-615.

[9] Lowe, D.G. (1995). "Similarity Metric Learning for a Variable-Kernel Classifier," *Neural Computation* **7**(1):72-85.

[10] Merz, C. and Murphy, P. (1996). UCI Repository of Machine Learning databases. http://www.ics.uci.edu/mlearn/MLRepository.html.

[11] Myles, J.P. and Hand, D.J. (1990). "The Multi-Class Metric Problem in Nearest Neighbor Discrimination Rules," *Pattern Recognition*, Vol. 23, pp. 1291-1297.

[12] Quinlin, J.R. (1993). *C4.5: Programs for Machine Learning*. Morgan-Kaufmann Publishers, Inc..
